# Single Transistor Learning Synapses

**Paul Hasler, Chris Diorio, Bradley A. Minch, Carver Mead**
California Institute of Technology
Pasadena, CA 91125
(818) 395 - 2812
paul@hobiecat.pcmp.caltech.edu

## Abstract

We describe single-transistor silicon synapses that compute, learn, and provide non-volatile memory retention. The single transistor synapses simultaneously perform long term weight storage, compute the product of the input and the weight value, and update the weight value according to a Hebbian or a backpropagation learning rule. Memory is accomplished via charge storage on polysilicon floating gates, providing long-term retention without refresh. The synapses efficiently use the physics of silicon to perform weight updates; the weight value is increased using tunneling and the weight value decreases using hot electron injection. The small size and low power operation of single transistor synapses allows the development of dense synaptic arrays. We describe the design, fabrication, characterization, and modeling of an array of single transistor synapses. When the steady state source current is used as the representation of the weight value, both the incrementing and decrementing functions are proportional to a power of the source current. The synaptic array was fabricated in the standard $2\mu m$ double - poly, analog process available from MOSIS.

## 1 INTRODUCTION

The past few years have produced a number of efforts to design VLSI chips which "learn from experience." The first step toward this goal is developing a silicon analog for a synapse. We have successfully developed such a synapse using only

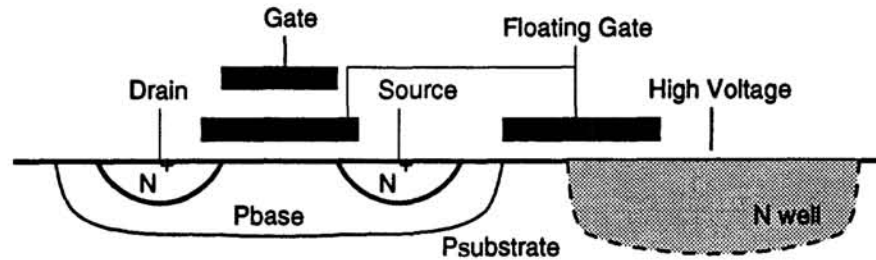

Figure 1: Cross section of the single transistor synapse. Our single transistor synapse uses a separate tunneling voltage terminal The pbase implant results in a larger threshold voltage, which results in all the electrons reaching the top of the $SiO_2$ barrier to be swept into the floating gate.

a single transistor. A synapse has two functional requirements. First, it must compute the product of the input multiplied by the strength (the weight) of the synapse. Second, the synapse must compute the weight update rule. For a Hebbian synapse, the change in the weight is the time average of the product of the input and output activity. In many supervised algorithms like backpropagation, this weight change is the time average of the product of the input and some fed back error signal. Both of these computations are similar in function. We have developed single transistor synapses which simultaneously perform long term weight storage, compute the product of the input and the weight value, and update the weight value according to a Hebbian or a backpropagation learning rule. The combination of functions has not previously been achieved with floating gate synapses.

There are five requirements for a learning synapse. First, the weight should be stored permanently in the absence of learning. Second, the synapse must compute as an output the product of the input signal with the synaptic weight. Third, each synapse should require minimal area, resulting in the maximum array size for a given area. Fourth, each synapse should operate with low power dissipation so that the synaptic array is not power constrained. And finally, the array should be capable of implementing either Hebbian or Backpropagation learning rule for modifying the weight on the floating gate. We have designed, fabricated, characterized, and modeled an array of single transistor synapses which satisfy these five criteria. We believe this is the first instance of a single transistor learning synapse fabricated in a standard process.

## 2  OVERVIEW

Figure 1 shows the cross section for the single transistor synapse. Since the floating gate is surrounded by $SiO_2$, an excellent insulator, charge leakage is negligible resulting in nearly permanent storage of the weight value. An advantage of using floating gate devices for learning rules is the timescales required to add and remove charge from the floating gate are well matched to the learning rates of visual and auditory signals. In addition, these learning rates can be electronically controlled. Typical resolution of charge on the floating gate after ten years is four bits (Holler 89). The FETs are in a moderately doped ($1 \times 10^{17} cm^{-3}$) substrate, to achieve a

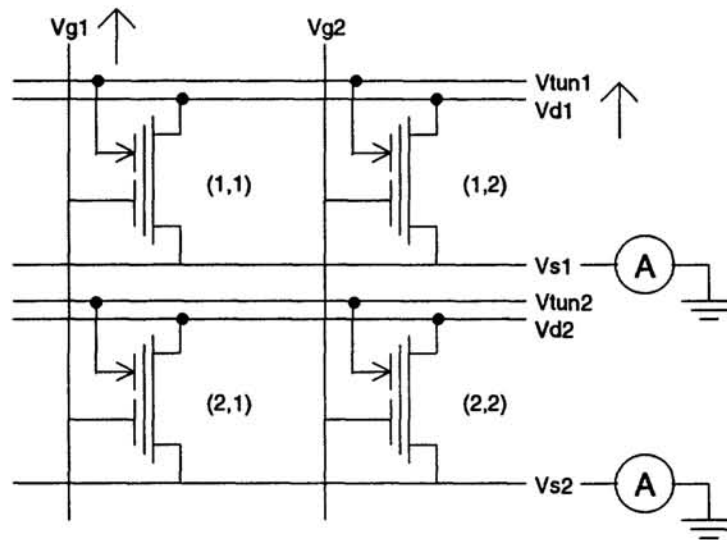

Figure 2: Circuit diagram of the single - transistor synapse array. Each transistor has a floating gate capacitively coupled to an input column line. A tunneling connection (arrow) allows weight increase. Weight decreased is achieved by hot electron injection in the transistor itself. Each synapse is capable of simultaneous feedforward computations and weight updates. A 2 x 2 section of the array allows us to characterize how modifying a single floating gate (such as synapse (1,1)) effects the neighboring floating gate values. The synapse currents are a measure of the synaptic weights, and are summed along each row by the source ($V_s$) or drain ($V_d$) lines into some soma circuit.

high threshold voltage. The moderately doped substrate is formed in the $2\mu m$ MOSIS process by the pbase implant. npn transistor. The implant has the additional benefit of increasing the efficiency of the hot electron injection process by increasing the electric field in the channel. Each synapse has an additional tunneling junction for modifying the charge on the floating gate. The tunneling junction is formed with high quality gate oxide separating a well region from the floating gate.

Each synapse in our synaptic array is a single transistor with its weight stored as a charge on a floating silicon gate. Figure 2 shows the circuit diagram of a 2 x 2 array of synapses. The column 'gate' inputs ($V_g$) are connected to second level polysilicon which capacitively couples to the floating gate. The inputs are shared along a column. The source ($V_s$), drain ($V_d$), and tunneling ($V_{tun}$) terminals are shared along a row. These terminals are involved with computing the output current and feeding back 'error' signal voltages. Many other synapses use floating gates to store the weight value, as in (Holler 89), but none of the earlier approaches update the charge on the floating gate during the multiplication of the input and floating gate value. In these previous approaches one must drive the floating gate over large a voltage range to tunnel electrons onto the floating gate. Synaptic computation must stop for this type of weight update.

The synapse computes as an output current a product of weight and input signal,

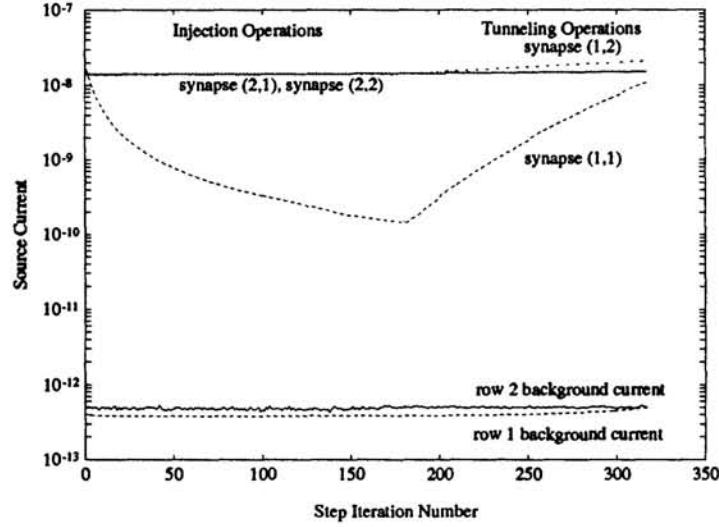

Figure 3: Output currents from a 2 x 2 section of the synapse array, showing 180 injection operations followed by 160 tunneling operations. For the injection operations, the drain $(Vd1)$ is pulsed from 2.0 V upto 3.3 V for 0.5s with $V_{g1}$ at 8V and $V_{g2}$ at $0V$. For the tunneling operations, the tunneling line $(Vtun1)$ is pulsed from 20 V up to 33.5 V with $V_{g2}$ at $0V$ and $V_{g1}$ at 8V. Because our measurements from the 2 x 2 section come from a larger array, we also display the 'background' current from all other synapses on the row. This background current is several orders of magnitude smaller than the selected synapse current, and therefore negligible.

and can simultaneously increment or decrement the weight as a function of its input and error voltages. The particular learning algorithm depends on the circuitry at the boundaries of the array; in particular the circuitry connected to each of the source, drain, and tunneling lines in a row. With charge $Q_{fg}$ on the floating gate and $V_s$ equal to 0 the subthreshold source current is described by

$$I_{synapse} = I_o e^{\frac{Q_{fg}}{Q_o}} e^{\frac{\delta V_g}{U_T}} \tag{1}$$

where $Q_o$ is a device dependent parameter, and $U_T$ is the thermal voltage $\frac{kT}{q}$. The coupling coefficient, $\delta$, of the gate input to the transistor surface potential is typically less than 0.1. From ( 1) We can consider the weight as a current $I$, defined by

$$I_{synapse} = \left( I_o e^{\frac{Q_{fg}}{Q_o}} e^{\frac{\delta V_{g0}}{U_T}} \right) e^{\frac{\delta \Delta V_g}{U_T}} = I e^{\frac{\delta \Delta V_g}{U_T}} \tag{2}$$

where $V_{g0}$ is the input voltage bias, and $\Delta V_g$ is $V_g - V_{g0}$. The synaptic current is thus the product of the weight, $I$, and a weak exponential function of the input voltage.

The single transistor learning synapses use a combination of electron tunneling and hot electron injection to adapt the charge on the floating gate, and thereby the

weight of the synapse. Hot electron injection adds electrons to the floating gate, thereby decreasing the weight. Injection occurs for large drain voltages; therefore the floating gate charge can be reduced during normal feedforward operation by raising the drain voltage. Electron tunneling removes electrons from the floating gate, thereby increasing the weight. The tunneling line controls the tunneling current; thus the floating gate charge can be increased during normal feedforward operation by raising the tunneling line voltage. The tunneling rate is modulated by both the input voltage and the charge on the floating gate.

Figure 3 shows an example the nature of the weight update process. The source current is used as a measure of the synapse weight. The experiment starts with all four synapses set to the same weight current. Then, synapse (1,1) is injected for 180 cycles to preferentially decrease its weight. Finally, synapse (1,1) is tunneled for 160 cycles to preferentially increase its weight. This experiment shows that a synapse can be incremented by applying a high voltage on tunneling terminals and a low voltage on the input, and can be decremented by applying a high voltage on drain terminals and a high voltage on the input. In the next two sections, we consider the nature of these update functions. In section three we examine the dependence of hot electron injection on the source current of our synapses. In section four we examine the dependence of electron tunneling on the source current of our synapses.

# 3  Hot Electron Injection

Hot electron injection gives us a method to add electrons to the floating gate. The underlying physics of the injection process is to give some electrons enough energy and direction in the channel to drain depletion region to surmount the $SiO_2$ energy barrier. A device must satisfy two requirements to inject an electron on a floating gate. First, we need a region where the potential drops more than 3.1 volts in a distance of less than $0.2\mu m$ to allow electrons to gain enough energy to surmount the oxide barrier. Second, we need a field in the oxide in the proper direction to collect electrons after they cross the barrier. The moderate substrate doping level allows us to easily achieve both effects in subthreshold operation. First, the higher substrate doping results in a much higher threshold voltage ($6.1V$), which guarantees that the field in the oxide at the drain edge of the channel will be in the proper direction for collecting electrons over the useful range of drain voltages. Second, the higher substrate doping results in higher electric fields which yield higher injection efficiencies. The higher injection efficiencies allow the device to have a wide range of drain voltages substantially below the threshold voltage. Figure 4 shows measured data on the change in source current during injection vs. source current for several values of drain voltage.

Because the source current, $I$, is related to the floating gate charge, $Q_{fg}$ as shown in ( 1) and the charge on the floating gate is related to the tunneling or injection current $(I_{fg})$ by

$$\frac{dQ_{fg}}{dt} = I_{fg} \qquad (3)$$

an approximate model for the change of the weight current value is

$$\frac{dI}{dt} = \frac{I}{Q_o}I_{fg} \qquad (4)$$

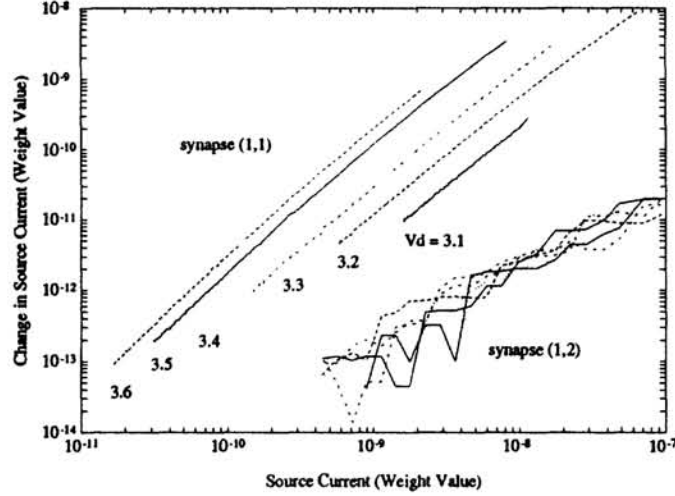

Figure 4: Source Current Decrement during injection vs. Source Current for several values of drain voltage. The injection operation decreases the synaptic weight. $Vg2$ was held at $0V$, and $Vg1$ was at $8V$ during the $0.5s$ injecting pulses. The change in source current is approximately proportional to the source current to the $\beta$ power, where of $\beta$ is between 1.7 and 1.85 for the range of drain voltages shown. The change in source current in synapse (1,2) is much less than the corresponding change in synapse (1,1) and is nearly independent of drain voltage. The effect of this injection on synapses (2,1) and (2,2) is negligible.

The injection current can be approximated over the range of drain voltages shown in Fig. 4 by (Hasler 95)

$$I_{fg} = -I_s e^{f(V_{d-c})} = -AI_s^{\beta-1} e^{\frac{V_d}{V_{inj}}} \tag{5}$$

where $V_{d-c}$ is the voltage from the drain to the drain edge of the channel, $V_d$ is the drain voltage, $f()$ is a slowly varying function defined in (Hasler 95), and $V_{inj}$ is in the range of $60mV$ to $100mV$. $A$ is device dependent parameter. Since hot electron injection adds electrons to the floating gate, the current into the floating gate $(I_{fg})$ is negative, which results in

$$\frac{dI}{dt} = -A\frac{I^\beta}{Q_o} e^{\frac{V_d}{V_{inj}}} \tag{6}$$

The model agrees well with the data in Fig. 4, with $\beta$ in the range of $1.7 - 1.9$. Injection is very selective along a row with a selectivity coefficient between $10^2$ and $10^7$ depending upon drain voltage and weight. The injection operations resulted in negligible changes in source current for synapses (2,1) and (2,2).

## 4    ELECTRON TUNNELING

Electron tunneling gives us a method for removing electrons from the floating gate. Tunneling arises from the fact that an electron wavefunction has finite extent. For a

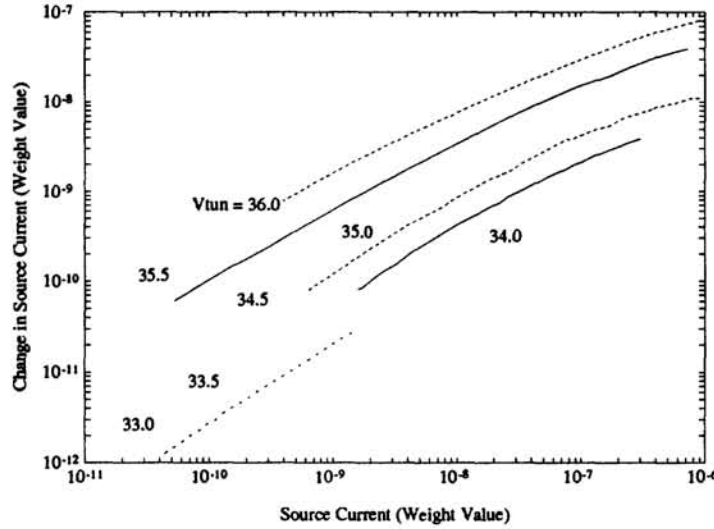

Figure 5: Synapse (1,1) source current increment vs. Source Current for several values of tunneling voltage. The tunneling operation increases the synaptic weight. $Vg1$ was held at $0V$ and $Vg2$ was $8V$ while the tunneling line was pulsed for $0.5s$ from $20V$ to the voltage shown. The change in source current is approximately proportional to the $\alpha$ power of the source current where $\alpha$ is between 0.7 and 0.9 for the range of tunneling voltages shown. The effect of this tunneling procedure on synapse (2,1) and (2,2) are negligible. The selectivity ratio of synapses on the same row is typically between 3-7 for our devices.

thin enough barrier, this extent is sufficient for an electron to penetrate the barrier. An electric field across the oxide will result in a thinner barrier to the electrons on the floating gate. For a high enough electric field, the electrons can tunnel through the oxide.

When traveling through the oxide, some electrons get trapped in the oxide, which changes the barrier profile. To reduce this trapping effect we tunnel through high quality gate oxide, which has far less trapping than interpoly oxide. Both injection and tunneling have very stable and repeatable characteristics. When tunneling at a fixed oxide voltage, the tunneling current decreases only 50 percent after $10nC$ of charge has passed through the oxide. This quantity of charge is orders of magnitude more than we would expect a synapse to experience over a lifetime of operation.

Figure 5 shows measured data on the change in source current during tunneling as a function of source current for several values of tunneling voltage. The functional form of tunneling current is of the form (Lenzlinger 69)

$$I_{fg} = I_{o_{tun}} e^{-\frac{V_o}{V_{tun} - V_{fg}}} \tag{7}$$

where $V_o$, $I_{o_{tun}}$ are model parameters which roughly correspond with theory. Tunneling removes electrons from the floating gate; therefore the floating gate current is positive. By expanding $V_{fg}$ for fixed $V_{tun}$ as $V_{fg0} + \Delta V_{fg}$ and inserting ( 1), the

| Parameter | Typical Values | Parameter | Typical Values |
|-----------|----------------|-----------|----------------|
| $\beta$ | 1.7 - 1.9 | $\alpha$ | 0.7 - 0.9 |
| $Q_o$ | .2 pC | $\delta$ | 0.02 - 0.1 |
| $A$ | $8.6 \times 10^{-20}$ | $V_{inj}$ | $78mV$ |

Table 1: Typical measured values of the parameters in the modeling of the single transistor synapse array.

resulting current change is

$$\frac{dI}{dt} = \frac{I_{0_{tun}}}{Q_o} e^{-\frac{V_o}{V_{tun}-V_{fg0}}} I_{s0} \left(\frac{I}{I_{s0}}\right)^{\alpha} \tag{8}$$

where $I_{s0}$ is the bias current corresponding to $V_{fg0}$. The model qualitatively agrees with the data in Fig. 5, with $\alpha$ in the range of 0.7 - 0.9. The tunneling selectivity between synapses on different rows is very good, but tunneling selectivity along along a row is poor. We typically measure tunneling selectivity ratios along a row between 3 - 7 for our devices.

## 5   Model of the Array of Single Transistor Synapses

Finally, we present an approximate model of our array of these single transistor synapses. The learning increment of the synapse at position $(i, j)$ can be modeled as

$$I_{synapse_{ij}} = I_{ij} e^{\frac{\delta \Delta V_g}{U_T}} \equiv I_{s0} W_{ij} x_j$$
$$\frac{dW_{ij}}{dt} = \frac{I_{0_{tun}}}{Q_o} e^{-\frac{V_o}{V_{tun}-V_{fg0}}} W_{ij}^{\alpha} x_j^{\alpha-1} - \frac{A}{Q_o} e^{\frac{V_{d_i}}{V_{inj}}} W_{ij}^{\beta} x_j^{\beta-1} \tag{9}$$

for the synapse at position $(i, j)$, where $W_{i,j}$ can be considered the weight value, and $x_j$ are the effective inputs network. Typical values for the parameters in ( 9 ) are given in Table 1.

## Acknowledgments

The work was supported by the office of Naval Research, the Advanced Research Projects Agency, and the Beckman Foundation.

## References

P. Hasler, C. Diorio, B. Minch, and C. Mead (1995) "An Analytic model of Hot Electron Injection from Boltzman Transport", *Tech. Report 123456*

M. Holler, S. Tam, H. Castro, and R. Benson (1989), "An electrically trainable artificial neural network with 10240 'floating gate' synapses", *International Joint Conference on Neural Networks, Washington, D.C., June 1989, pp. II-191 - II-196.*

M. Lenzlinger and E. H. Snow (1969), "Fowler-Nordheim tunneling into thermally grown $SiO_2$," *J. Appl. Phys., vol. 40, pp. 278-283, 1969.*
